# On Tracking The Partition Function

**Guillaume Desjardins, Aaron Courville, Yoshua Bengio**
{desjagui,courvila,bengioy}@iro.umontreal.ca
Département d'informatique et de recherche opérationnelle
Université de Montréal

## Abstract

Markov Random Fields (MRFs) have proven very powerful both as density estimators and feature extractors for classification. However, their use is often limited by an inability to estimate the partition function $Z$. In this paper, we exploit the gradient descent training procedure of restricted Boltzmann machines (a type of MRF) to **track** the log partition function during learning. Our method relies on two distinct sources of information: (1) estimating the change $\Delta Z$ incurred by each gradient update, (2) estimating the difference in $Z$ over a small set of tempered distributions using bridge sampling. The two sources of information are then combined using an inference procedure similar to Kalman filtering. Learning MRFs through Tempered Stochastic Maximum Likelihood, we can estimate $Z$ using no more temperatures than are required for learning. Comparing to both exact values and estimates using annealed importance sampling (AIS), we show on several datasets that our method is able to accurately track the log partition function. In contrast to AIS, our method provides this estimate at each time-step, at a computational cost similar to that required for training alone.

## 1 Introduction

In many areas of application, problems are naturally expressed as a Gibbs measure, where the distribution over the domain $\mathcal{X}$ is given by, for $x \in \mathcal{X}$:

$$q(x) = \frac{\tilde{q}(x)}{Z(\beta)} = \frac{\exp\{-\beta E(x)\}}{Z(\beta)}, \text{ with } Z(\beta) = \sum_x \tilde{q}(x). \tag{1}$$

$E(x)$ is refered to as the "energy" of configuration $x$, $\beta$ is a free parameter known as the inverse temperature and $Z(\beta)$ is the normalization factor commonly refered to as the partition function. Under certain general conditions on the form of $E$, these models are known as Markov Random Fields (MRF), and have been very popular within the vision and natural language processing communities. MRFs with latent variables – in particular restricted Boltzmann machines (RBMs) [9] – are among the most popular building block for deep architectures [1], being used in the unsupervised initialization of both Deep Belief Networks [9] and Deep Boltzmann Machines [22].

As illustrated in Eq. 1, the partition function is computed by summing over all variable configurations. Since the number of configurations scales exponentially with the number of variables, exact calculation of the partition function is generally computationally intractable. Without the partition function, probabilities under the model can only be determined up to a multiplicative constant, which seriously limits the model's utility. One method recently proposed for estimating $Z(\beta)$ is annealed importance sampling (AIS) [18, 23]. In AIS, $Z(\beta)$ is approximated by the sum of a set of importance-weighted samples drawn from the model distribution. With a large number of variables, drawing a set of importance-weighted samples is generally subject to extreme variance in the importance weights. AIS alleviates this issue by annealing the model distribution through a series of slowly changing distributions that link the target model distribution to one where the log partition function is tractable. While AIS is quite successful, it generally requires the use of tens of thousands of annealing distributions in order to achieve accurate results. This computationally intensive

requirement renders AIS inappropriate as a means of maintaining a running estimate of the log partition function throughout training. Yet, having ready access to this quantity throughout learning opens the door to a range of possibilities. Likelihood could be used as a basis for model comparison throughout training; early-stopping could be accomplished by monitoring an estimate of the likelihood of a validation set. Another important application is in Bayesian inference in MRFs [17] where we require the partition function for each value of the parameters in the region of support. Tracking the log partition function would also enable simultaneous estimation of all the parameters of a heterogeneous model, for example an extended directed graphical model with Gibbs distributions forming some of the model components.

In this work, we consider a method of *tracking* the log partition function during training, which builds upon the parallel tempering (PT) framework [7, 10, 15]. Our method relies on two basic observations. First, when using stochastic gradient descent [1], parameters tend to change slowly during training; consequently, the partition function $Z(\beta)$ also tends to evolve slowly. We exploit this property of the learning process by using importance sampling to estimate changes in the log partition function from one learning iteration the next. If the changes in the distribution from time-step $t$ to $t + 1$ are small, the importance sampling estimate can be very accurate, even with relatively few samples. This is the same basic strategy employed in AIS, but while with AIS one constructs a path of close distributions through an annealing schedule, in our procedure we simply rely on the path of distributions that emerges from the learning process. Second, parallel tempering (PT) relies on simulating an extended system, consisting of multiple models each running at their own temperature. These temperatures are chosen such that neighboring models overlap sufficiently as to allow for frequent cross-temperature state swaps. This is an ideal operating regime for bridge sampling [2, 19], which can thus serve to estimate the difference in log partition functions between neighboring models. While with relatively few samples, each method on its own tends not to provide reliable estimates, we propose to combine these measurements using a variation of the well-known Kalman filter (KF), allowing us to accurately track the evolution of the log partition function throughout learning. The efficiency of our method stems from the fact that our estimator makes use of the samples generated in the course of training, thus incurring relatively little additional computational cost.

This paper is structured as follows. In Section 2, we provide a brief overview of RBMs and the SML-PT training algorithm, which serves as the basis of our tracking algorithm. Sections (3.1-3.3) cover the details of the importance and bridge sampling estimates, while Section 3.4 provides a comprehensive look at our filtering procedure and the tracking algorithm as a whole. Experimental results are presented in Section 4.

## 2 Stochastic Maximum Likelihood with Parallel Tempering

Our proposed log partition function tracking strategy is applicable to any Gibbs distribution model that is undergoing relatively smooth changes in the partition function. However, we concentrate on its application to the RBM since it has become a model of choice for learning unsupervised features for use in deep feed-forward architectures [9, 1] as well as for modeling complex, high-dimensional distributions [27, 24, 12].

RBMs are bipartite graphical models where visible units $\mathbf{v} \in \{0, 1\}^{n_v}$ interact with hidden units $\mathbf{h} \in \{0, 1\}^{n_h}$ through the energy function $E(\mathbf{v}, \mathbf{h}) = -\mathbf{h}^T W \mathbf{v} - c^T \mathbf{h} - b^T \mathbf{v}$. The model parameters $\theta = [W, c, b]$ consist of the weight matrix $W \in \mathcal{R}^{n_h \times n_v}$, whose entries $W_{ij}$ connect units $(v_i, h_j)$, and offset vectors $b$ and $c$. RBMs can be trained through a stochastic approximation to the negative log-likelihood gradient $\frac{\partial F(\mathbf{v})}{\partial \theta} - \mathbb{E}_p[\frac{\partial F(\mathbf{v})}{\partial \theta}]$, where $F(v)$ is the free-energy function defined as $F(\mathbf{v}) = -\log \sum_h \exp(-E(\mathbf{v}, \mathbf{h}))$. In Stochastic Maximum Likelihood (SML) [25], we replace the expectation by a sample average, where approximate samples are drawn from a persistent Markov chain, updated through $k$-steps of Gibbs sampling between parameter updates. Other algorithms improve upon this default formulation by replacing Gibbs sampling with more powerful sampling algorithms [26, 7, 21, 20]. By increasing the mixing rate of the underlying Markov chain, these methods can lead to lower variance estimates of the maximum likelihood gradient and faster conver-

gence. However, from the perspective of tracking the log partition function, we will see in Section 3 that the SML-PT scheme [7] presents a rather unique advantage.

Throughout training, parallel tempering draws samples from an extended system $\mathcal{M}_t = \{q_{i,t}; i \in [1, M]\}$, where $q_{i,t}$ denotes the model with inverse temperature $\beta_i \in [0, 1]$ obtained after $t$ steps of gradient descent. Each model $q_{i,t}$ (associated with a unique partition function $Z_{i,t}$) represents a smoothed version of the target distribution: $q_{1,t}$ (with $\beta_1 = 1$). The inverse temperature $\beta_i = 1/T_i \in [0, 1]$ controls the degree of smoothing, with smaller values of $\beta_i$ leading to distributions which are easier to sample from. To leverage these fast-mixing chains, PT alternates $k$ steps of Gibbs sampling (performed independently at each temperature) with cross-temperature state swaps. These are proposed between neighboring chains using a Metropolis-Hastings-based acceptance criterion. If we denote the particle obtained by each model $q_{i,t}$ after $k$ steps of Gibbs sampling as $x_{i,t}$, then the swap acceptance ratio $r_{i,t}$ for chains $(i, i + 1)$ is given by:

$$r_{i,t} = \min\left(1, \frac{\tilde{q}_{i,t}(\mathbf{x}_{i+1,t})\tilde{q}_{i+1,t}(\mathbf{x}_{i,t})}{\tilde{q}_{i,t}(\mathbf{x}_{i,t})\tilde{q}_{i+1,t}(\mathbf{x}_{i+1,t})}\right) \tag{2}$$

These swaps ensure that samples from highly ergodic chains are gradually swapped into lower temperature chains. Our swapping schedule is the deterministic even-odd algorithm [14] which proposes swaps between all pairs $(q_{i,t}, q_{i+1,t})$ with even $i$'s, followed by those with odd $i$'s. The gradient is then estimated by using the sample which was last swapped into temperature $\beta_1$. To reduce the variance on our estimate, we run multiple Markov chains per temperature, yielding a mini-batch of model samples $\mathcal{X}_{i,t} = \{x_{i,t}^{(n)} \sim q_{i,t}(x); 1 \leq n \leq N\}$ at each time-step and temperature.

SML with Adaptive parallel tempering (SML-APT) [6], further improves upon SML-PT by automating the choice of temperatures. It does so by maximizing the flow of particles between extremal temperatures, yielding better ergodicity and more robust sampling in the negative phase of training.

## 3   Tracking the Partition Function

Unrolling in time (learning iterations) the $M$ models being simulated by PT, we can envision a two-dimensional lattice of RBMs indexed by $(i, t)$. As previously mentioned, gradient descent learning causes $q_{i,t}$, the model with inverse temperature $\beta_i$ obtained at time-step $t$, to be close to $q_{i,t-1}$. We can thus apply importance sampling between adjacent temporal models [2] to obtain an estimate of $\zeta_{i,t} - \zeta_{i,t-1}$, denoted as $O_{i,t}^{\Delta t}$. Inspired by the annealing distributions used in AIS, one could think to iterate this process from a known quantity $\zeta_{i,1}$, in order to estimate $\zeta_{i,t}$. Unfortunately, the variance of such an estimate would grow quickly with $t$.

PT provides an interesting solution to this problem, by simulating an extended system $\mathcal{M}_t$ where the $\beta_i$'s are selected such that $q_{i,t}$ and $q_{i+1,t}$ have enough overlap to allow for frequent cross-temperature state swaps. This motivates using bridge sampling [2] to provide an estimate of $\zeta_{i+1,t} - \zeta_{i,t}$, the difference in log partitions between temperatures $\beta_{i+1}$ and $\beta_i$. We denote this estimate as $O_{i,t}^{\Delta\beta}$. Additionally, we can treat $\zeta_{M,t}$ as a known quantity during training, by setting $\beta_M = 0$ [3]. Beginning with $\zeta_{M,t}$ (see definition in Fig. 1), repeated application of bridge sampling alone could in principle arrive at an accurate estimate of $\{\zeta_{i,t}; i \in [1, M], t \in [1, T]\}$. However, reducing the variance sufficiently to provide useful estimates of the log partition function would require using a relatively large number of samples at each temperature. Within the context of RBM training, the required number of samples at *each* of the parallel chains would have an excessive computational cost. Nonetheless even with relatively few samples, the bridge sampling estimate provides an additional source of information regarding the log partition function.

Our strategy is to combine these two high variance estimates $O_{i,t}^{\Delta t}$ and $O_{i,t}^{\Delta\beta}$ by treating the unknown log partition functions as a latent state to be tracked by a Kalman filter. In this framework, we consider $O_{i,t}^{\Delta t}$ and $O_{i,t}^{\Delta\beta}$ as observed quantities, used to iteratively refine the joint distribution over the latent state at each learning iteration. Formally, we define this latent state to be $\zeta_t = [\zeta_{1,t}, \ldots, \zeta_{M,t}, b_t]$, where $b_t$ is an extra term to account for a systematic bias in $O_{1,t}^{\Delta\beta}$ (see Sec. 3.2 for details). The corresponding graphical model is shown in Figure 1.

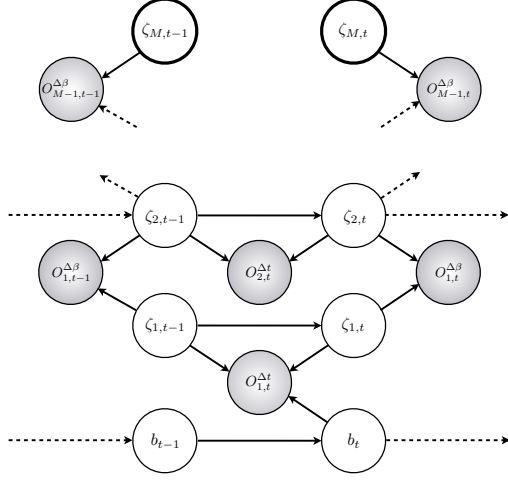

System Equations:

$$p(\zeta_0) = \mathcal{N}(\mu_0, \Sigma_0)$$

$$p(\zeta_t \mid \zeta_{t-1}) = \mathcal{N}(\zeta_{t-1}, \Sigma_\zeta)$$

$$p(O_t^{(\Delta t)} \mid \zeta_t, \zeta_{t-1}) = \mathcal{N}(C[\zeta_t, \zeta_{t-1}]^T, \ \Sigma_{\Delta t})$$

$$p(O_t^{(\Delta \beta)} \mid \zeta_t) = \mathcal{N}(H\zeta_t, \Sigma_{\Delta \beta})$$

$$C = \begin{bmatrix} & 1 & & 0 \\ & 0 & & 0 \\ I_M & \vdots & -I_M & \vdots \\ & 0 & & 0 \end{bmatrix}$$

$$H = \begin{bmatrix} -1 & +1 & 0 & 0 & & 0 \\ 0 & -1 & +1 & 0 & \vdots & 0 \\ & & \cdots & & & 0 \\ 0 & 0 & 0 & -1 & +1 & 0 \end{bmatrix}$$

Figure 1: A directed graphical model for log partition function tracking. The shaded nodes represent observed variables, and the double-walled nodes represent the tractable $\zeta_{M,:}$ with $\beta_M = 0$. For clarity of presentation, we show the bias term as distinct from the other $\zeta_{i,t}$ (recall $b_t = \zeta_{M+1,t}$.)

## 3.1 Model Dynamics

The first step is to specify how we expect the log partition function to change over training iterations, i.e. our prior over the model dynamics. SML training of the RBM model parameters is a stochastic gradient descent algorithm (typically over a mini-batch of $N$ examples) where the parameters change by small increments specified by an approximation to the likelihood gradient. This implies that both the model distribution and the partition function change relatively slowly over learning increments with a rate of change being a function of the SML learning rate; i.e. we expect $q_{i,t}$ and $\zeta_{i,t}$ to be close to $q_{i,t-1}$ and $\zeta_{i,t-1}$ respectively.

Our model dynamics are thus simple and capture the fact that the log partition function is slowly changing. Characterizing the evolution of the log partition functions as independent Gaussian processes, we model the probability of $\zeta_t$ conditioned on $\zeta_{t-1}$ as $p(\zeta_t|\zeta_{t-1}) = \mathcal{N}(\zeta_{t-1}, \Sigma_\zeta)$, a normal distribution with mean $\zeta_{t-1}$ and fixed diagonal covariance $\Sigma_\zeta = \text{Diag}[\sigma_Z^2, \ldots, \sigma_Z^2, \sigma_b^2]$. $\sigma_Z^2$ and $\sigma_b^2$ are hyper-parameters controlling how quickly the latent states $\zeta_{i,t}$ and $b_t$ are expected to change between learning iterations.

## 3.2 Importance Sampling Between Learning Iterations

The observation distribution $p(O_t^{(\Delta t)} \mid \zeta_t, \zeta_{t-1}) = \mathcal{N}(C[\zeta_t, \zeta_{t-1}]^T, \Sigma_{\Delta t})$ models the relationship between the evolution of the latent log partitions and the statistical measurements $O_t^{(\Delta t)} = [O_{1,t}^{(\Delta t)}, \ldots, O_{M,t}^{(\Delta t)}]$ given by importance sampling, with $O_{i,t}^{\Delta t}$ defined as:

$$O_{i,t}^{\Delta t} = \log \left\{ \frac{1}{N} \sum_{n=1}^{N} w_{i,t}^{(n)} \right\} \quad \text{with} \quad w_{i,t}^{(n)} = \frac{\tilde{q}_{i,t}(x_{i,t-1}^{(n)})}{\tilde{q}_{i,t-1}(x_{i,t-1}^{(n)})}. \tag{3}$$

In the above distribution, the matrix $C$ encodes the fact that the average importance weights estimate $\zeta_{i,t} - \zeta_{i,t-1} + b_t \cdot I_{i=1}$, where $I$ is the indicator function. It is formally defined in Fig. 1. $\Sigma_{\Delta t}$ is a diagonal covariance matrix, whose elements are updated online from the estimated variances of the log-importance weights. At time-step $t$, the $i$-th entry of its diagonal is thus given by $\text{Var}[w_{i,t}]/\left[\sum_n w^{(n)}\right]^2$.

The term $b_t$ accounts for a systematic bias in $O_{1,t}^{(\Delta t)}$. It stems from the reuse of samples $\mathcal{X}_{1,t-1}$: first, for estimating the negative phase gradient at time-step $t-1$ (i.e. the gradient applied between $q_{i,t-1}$

and $q_{i,t}$) and second, to compute the importance weights of Eq. 3. Since the SML gradient acts to lower the probability of negative particles, $w_{i,t}^{(n)}$ is biased.

## 3.3 Bridging the Parallel Tempering Temperature Gaps

Consider now the other dimension of our parallel tempered lattice of RBMs: temperature. As previously mentioned, neighboring distributions in PT are designed to have significant overlap in their densities in order to permit particle swaps. However, the intermediate distributions $q_{i,t}(\mathbf{v}, \mathbf{h})$ are not so close to one another that we can use them as the intermediate distributions of AIS. AIS typically requires thousands of intermediate chains, and maintaining that number of parallel chains would carry a prohibitive computational burden. On the other hand, the parallel tempering strategy of spacing the temperature to ensure moderately frequent swapping nicely matches the ideal operating regime of bridge sampling [2].

We thus consider a second observation model as $p(O_t^{(\Delta\beta)} \mid \zeta_t) = \mathcal{N}(H\zeta_t, \Sigma_{\Delta\beta})$, with $H$ defined in Fig.1. The quantities $O_t^{(\Delta\beta)} = [O_{1,t}^{\Delta\beta}, \dots, O_{M-1,t}^{\Delta\beta}]$ are obtained via bridge sampling as estimates of $\zeta_{i+1,t} - \zeta_{i,t}$. Entries $O_{i,t}^{\Delta\beta}$ are given by:

$$O_{i,t}^{\Delta\beta} = \log \sum_{n=1}^{N} u_{i,t}^{(n)} - \log \sum_{n=1}^{N} v_{i,t}^{(n)}, \text{ where } u_{i,t}^{(n)} = \frac{q_{i,t}^*\left(x_{i,t}^{(n)}\right)}{\tilde{q}_{i,t}\left(x_{i,t}^{(n)}\right)}, \quad v_{i,t}^{(n)} = \frac{q_{i,t}^*\left(x_{i+1,t}^{(n)}\right)}{\tilde{q}_{i+1,t}\left(x_{i+1,t}^{(n)}\right)}. \quad (4)$$

The bridging distribution [2, 19] $q_{i,t}^*$ is chosen such that it has large support with both $q_i$ and $q_{i+1}$. For all $i \in [1, M-1]$, we choose the approximately optimal distribution $q_{i,t}^{(opt)}(x) = \frac{\tilde{q}_{i,t}(x)\tilde{q}_{i+1,t}(x)}{s_{i,t}\tilde{q}_{i,t}(x) + \tilde{q}_{i+1,t}(x)}$ where $s_{i,t} \approx Z_{i+1,t}/Z_{i,t}$. Since the $Z_{i,t}$'s are the very quantities we are trying to estimate, this definition may seem problematic. However it is possible to start with a coarse estimate of $s_{i,1}$ and refine it in subsequent iterations by using the output of our tracking algorithm. $\Sigma_{\Delta\beta}$ is once again a diagonal covariance matrix, updated online from the variance of the log-importance weights $u$ and $v$ [19]. The $i$-th entry is given by $\frac{\text{Var}[u_{i,t}]}{\left[\sum_n u_{i,t}^{(n)}\right]^2} + \frac{\text{Var}[v_{i,t}]}{\left[\sum_n v_{i,t}^{(n)}\right]^2}$.

## 3.4 Kalman Filtering of the Log-Partition Function

In the above we have described two sources of information regarding the log partition function for each of the RBMs in the lattice. In this section we describe a method to fuse all available information to improve the overall accuracy of the estimate of every log partition function. We now consider the steps involved in the inference process in moving from an estimate of the posterior over the latent state at time $t-1$ to an estimate of the posterior at time $t$. We begin by assuming we know the posterior $p(\zeta_{t-1} \mid O_{t-1:0}^{(\Delta t)}, O_{t-1:0}^{(\Delta\beta)})$, where $O_{t-1:0}^{(\cdot)} = [O_1^{(\cdot)}, \dots, O_{t-1}^{(\cdot)}]$.

We follow the treatment of Neal [18] in characterizing our uncertainty regarding $\zeta_{i,t}$ as a Gaussian distribution and define $p(\zeta_{t-1} \mid O_{t-1:0}^{(\Delta t)}, O_{t-1:0}^{(\Delta\beta)}) \sim \mathcal{N}(\mu_{t-1,t-1}, P_{t-1,t-1})$, a multivariate Gaussian with mean $\mu_{t-1,t-1}$ and covariance $P_{t-1,t-1}$. The double index notation is used to indicate which is the latest observation being conditioned on for each of the two types of observations: e.g. $\mu_{t,t-1}$ represents the posterior mean given $O_{t:0}^{(\Delta t)}$ and $O_{t-1:0}^{(\Delta\beta)}$.

Departing from the typical Kalman filter setting, $O_t^{(\Delta t)}$ depends on both $\zeta_t$ and $\zeta_{t-1}$. In order to incorporate this observation into our estimate of the latent state, we first need to specify the prior joint distribution $p(\zeta_{t-1}, \zeta_t \mid O_{t-1:0}^{(\Delta t)}, O_{t-1:0}^{(\Delta\beta)}) = p(\zeta_t \mid \zeta_{t-1})p(\zeta_{t-1} \mid O_{t-1:0}^{(\Delta t)}, O_{t-1:0}^{(\Delta\beta)})$, with $p(\zeta_t \mid \zeta_{t-1})$ as defined in Sec. 3.1. Observation $O_t^{(\Delta t)}$ is then incorporated through Bayes rule, yielding $p(\zeta_{t-1}, \zeta_t \mid O_{t:0}^{(\Delta t)}, O_{t-1:0}^{(\Delta\beta)})$. Having incorporated the importance sampling estimate into the model, we can then marginalize over $\zeta_{t-1}$ (which is no longer required), to yield $p(\zeta_t \mid O_{t:0}^{(\Delta t)}, O_{t-1:0}^{(\Delta\beta)})$. Finally, it remains only to incorporate the bridge sampler estimate $O_t^{(\Delta\beta)}$ by a second application of Bayes rule, which gives us $p(\zeta_t \mid O_{t:0}^{(\Delta t)}, O_{t:0}^{(\Delta\beta)})$, the updated posterior over the latent state at time-step $t$. The detailed inference equations are provided in Fig. 2 and can be derived easily from standard textbook equations on products and marginals of normal distributions [4].

Inference Equations:

(i) $\quad p\left(\zeta_{t-1}, \zeta_t \mid O_{t-1:0}^{(\Delta t)}, O_{t-1:0}^{(\Delta \beta)}\right) = \mathcal{N}(\eta_{t-1,t-1}, V_{t-1,t-1})$

$\quad$ with $\eta_{t-1,t-1} = \begin{bmatrix} \mu_{t-1,t-1} \\ \mu_{t-1,t-1} \end{bmatrix}$ and $V_{t-1,t-1} = \begin{bmatrix} P_{t-1,t-1} & P_{t-1,t-1} \\ P_{t-1,t-1} & \Sigma_\zeta + P_{t-1,t-1} \end{bmatrix}$

(ii) $\quad p(\zeta_{t-1}, \zeta_t \mid O_{t:0}^{(\Delta t)}, O_{t-1:0}^{(\Delta \beta)}) = \mathcal{N}(\eta_{t,t-1}, V_{t,t-1})$

$\quad$ with $V_{t,t-1} = (V_{t-1,t-1}^{-1} + C^T \Sigma_{\Delta t}^{-1} C)^{-1}$ and $\eta_{t,t-1} = V_{t,t-1}(C^T \Sigma_{\Delta t} O_t^{(\Delta t)} + V_{t-1,t-1}^{-1} \eta_{t-1,t-1})$

(iii) $\quad p\left(\zeta_t \mid O_{t:0}^{(\Delta t)}, O_{t-1:0}^{(\Delta \beta)}\right) = \mathcal{N}(\mu_{t,t-1}, P_{t,t-1})$ with $\mu_{t,t-1} = [\eta_{t,t-1}]_2$ and $P_{t,t-1} = [V_{t,t-1}]_{2,2}$

(iv) $\quad p(\zeta_t \mid O_{t:0}^{(\Delta t)}, O_{t:0}^{(\Delta \beta)}) = \mathcal{N}(\mu_{t,t}, P_{t,t})$

$\quad$ with $P_{t,t} = (P_{t,t-1}^{-1} + H^T \Sigma_{\Delta \beta}^{-1} H)^{-1}$ and $\mu_{t,t} = P_{t,t}(H^T \Sigma_{\Delta \beta} O_t^{(\Delta \beta)} + P_{t,t-1}^{-1} \mu_{t,t-1})$

Figure 2: Inference equations for our log partition tracking algorithm, a variant on the Kalman filter. For any vector $v$ and matrix $V$, we use the notation $[v]_2$ to denote the vector obtained by preserving the bottom half elements of $v$ and $[V]_{2,2}$ to indicate the lower right-hand quadrant of $V$.

## 4 Experimental Results

For the following experiments, SML was performed using either constant or decreasing learning rates. We used the decreasing schedule $\epsilon_t = \min(\epsilon_{\text{init}} \frac{\alpha}{t+1}, \epsilon_{\text{init}})$, where $\epsilon_t$ is the learning rate at time-step $t$, $\epsilon_{\text{init}}$ is the initial or base learning rate and $\alpha$ is the decrease constant. Entries of $\Sigma_\zeta$ (see Section 3.1) were set as follows. We set $\sigma_Z^2 = +\infty$, which is to say that we did not exploit the smoothness prior when estimating the prior distribution over the joint $p(\zeta_{t-1}, \zeta_t \mid O_{t-1:0}^{(\Delta t)}, O_{t-1:0}^{(\Delta \beta)})$. $\sigma_b^2$ was set to $10^{-3} \cdot \epsilon_t$, allowing the estimated bias on $O_{1,t}^{(\Delta t)}$ to change faster for large learning rates.

When initializing the RBM visible offsets[4] as proposed in [8], the intermediate distributions of Eq. 1 lead to sub-optimal swap rates between adjacent chains early in training, with a direct impact on the quality of tracking. In our experiments, we avoid this issue by using the intermediate distributions $q_{i,t}(x) \propto \exp[\beta_i \cdot (-\mathbf{h}^T W \mathbf{v} - c^T \mathbf{h}) - b^T \mathbf{v}]$. We tested mini-batch sizes $N \in [10, 20]$.

**Comparing to Exact Likelihood** We start by comparing the performance of our tracking algorithm to the exact likelihood, obtained by marginalizing over both visible and hidden units. We chose 25 hidden units and trained on the ubiquitous MNIST [13] dataset for 300k updates, using both fixed and adaptive learning rates. The main results are shown in Figure 3.

In Figure 3(a), we can see that our tracker provides a very good fit to the likelihood with $\epsilon_{\text{init}} = 0.001$ and decrease constants $\alpha$ in $\{10^3, 10^4, 10^5\}$. Increasing the base learning rate to $\epsilon_{\text{init}} = 0.01$ in Figure 3(b), we maintain a good fit up to $\alpha = 10^4$, with a small dip in performance at 50k updates. Our tracker fails however to capture the oscillatory behavior engendered by too high of a learning rate ($\epsilon_{\text{init}} = 0.01, \alpha = 10^5$). It is interesting to note that the failure mode of our algorithm seems to coincide with an unstable optimization process.

**Comparing to AIS for Large-Scale Models** In evaluating the performance of our tracking algorithm on larger models, exact computation of the likelihood is no longer possible, so we use AIS as our baseline.[5] Our models consisted of RBMs with 500 hidden units, trained using SML-APT [6] on the MNIST and Caltech Silhouettes [16] datasets. We performed 200k updates, with learning rate parameters $\epsilon_{\text{init}} \in \{.01, .001\}$ and $\alpha \in \{10^3, 10^4, 10^5\}$.

On MNIST, AIS estimated the test-likelihood of our best model at $-94.34 \pm 3.08$ (where $\pm$ indicates the $3\sigma$ confidence interval), while our tracking algorithm reported a value $-89.96$. On Caltech Silhouettes, our model reached $-134.23 \pm 21.14$ according to AIS, while our tracker reported

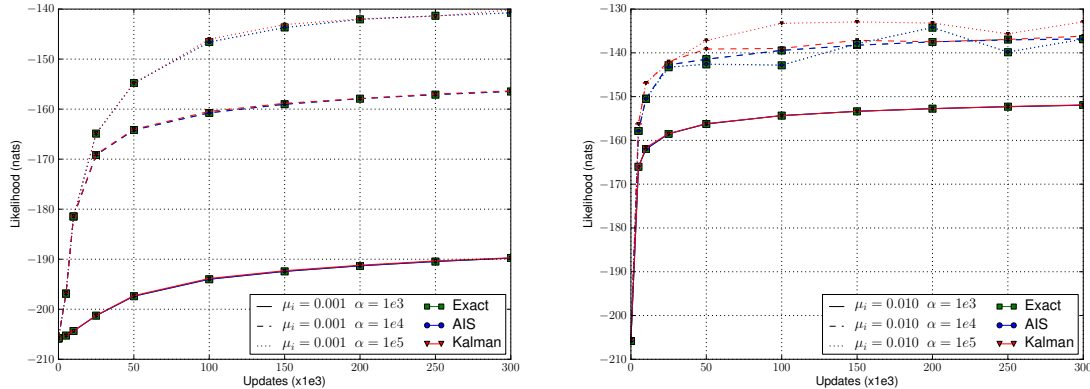

Figure 3: Comparison of exact test-set likelihood and estimated likelihood as given by AIS and our tracking algorithm. We trained a 25-hidden unit RBM for 300k updates using SML, with a learning rate schedule $\epsilon_t = \min(\alpha \cdot \epsilon_{init}/(t+1), \epsilon_{init})$, with (left) $\epsilon_{init} = 0.001$ and (right) $\epsilon_{init} = 0.01$ varying $\alpha \in \{10^3, 10^4, 10^5\}$.

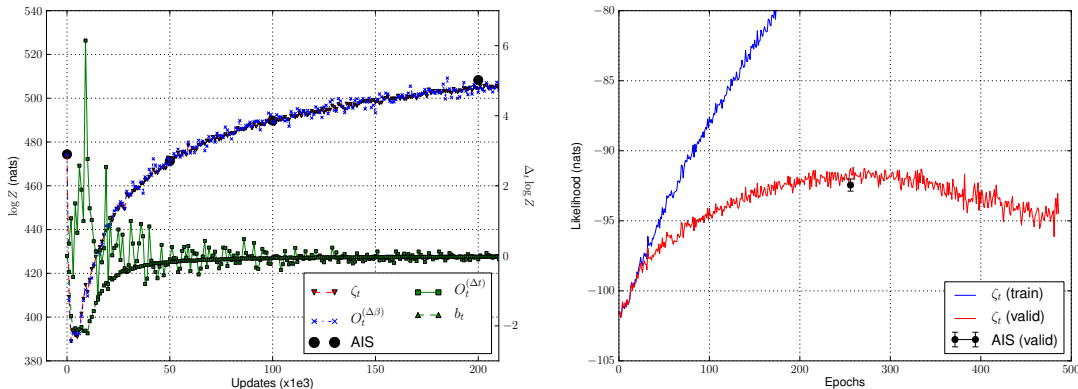

Figure 4: (left) Plotted on left y-axis are the Kalman filter measurements $O_t^{(\Delta\beta)}$, our log partition estimate of $\zeta_{1,t}$ and point estimates of $\zeta_{1,t}$ obtained by AIS. On the right y-axis, measurement $O_t^{(\Delta t)}$ is plotted, along with the estimated bias $b_t$. Note how $b_t$ becomes progressively less-pronounced as $\epsilon_t$ decreases and the model converges. Also of interest, the variance on $O_t^{(\Delta\beta)}$ increases with $t$ but is compensated by a decreasing variance on $O_t^{(\Delta t)}$, yielding a relatively smooth estimate $\zeta_{1,t}$. (not shown) The $\pm 3\sigma$ confidence interval of the AIS estimate at 200k updates was measured to be 3.08. (right) Example of early-stopping on dna dataset.

$-114.31$. To put these numbers in perspective, Salakhutdinov and Murray [23] reports values of $-125.53$, $-105.50$ and $-86.34$ for 500 hidden unit RBMs trained with CD\{1,3,25\} respectively. Marlin *et al.* [16] report around $-120$ for Caltech Silhouettes, again using 500 hidden units.

Figure 4(left) shows a detailed view of the Kalman filter measurements and its output, for the best performing MNIST model. We can see that the variance on $O_t^{(\Delta\beta)}$ (plotted on the left y-axis) grows slowly over time, which is mitigated by a decreasing variance on $O_t^{(\Delta t)}$ (plotted on the right y-axis). As the model converges and the learning rate decreases, $q_{i,t-1}$ and $q_{i,t}$ become progressively closer and the importance sampling estimates become more robust. The estimated bias term $b_t$ also converges to zero.

An important point to note is that a naive linear-spacing of temperatures yielded low exchange rates between neighboring temperatures, with adverse effects on the quality of our bridge sampling estimates. As a result, we observed a drop in performance, both in likelihood as well as tracking performance. Adaptive tempering [6] (with a fixed number of chains $M$) proved crucial in getting good tracking for these experiments.

**Early-Stopping Experiments** Our final set of experiments highlights the performance of our method on a wide-variety of datasets [11]. In these experiments, we use our estimate of the log

| Dataset | RBM | | | RBM-25 | NADE |
|---|---|---|---|---|---|
| | Kalman | AIS | | | |
| adult | -15.24 | -15.70 | ($\pm$ 0.50) | -16.29 | -13.19 |
| connect4 | -15.77 | -16.81 | ($\pm$ 0.67) | -22.66 | -11.99 |
| dna | -87.97 | -88.51 | ($\pm$ 0.97) | -96.90 | -84.81 |
| mushrooms | -10.49 | -14.68 | ($\pm$ 30.75) | -15.15 | -9.81 |
| nips | -270.10 | -271.23 | ($\pm$ 0.58) | -277.37 | -273.08 |
| ocr_letters | -33.87 | -31.45 | ($\pm$ 2.70) | -43.05 | -27.22 |
| rcv1 | -46.89 | -48.61 | ($\pm$ 0.69) | -48.88 | -46.66 |
| web | -28.95 | -29.91 | ($\pm$ 0.74) | -29.38 | -28.39 |

Table 1: Test set likelihood on various datasets. Models were trained using SML-PT. Early-stopping was performed by monitoring likelihood on a hold-out validation set, using our KF estimate of the log partition function. Best models (i.e. the choice of hyper-parameters) were then chosen according to the AIS likelihood estimate. Results for 25-hidden unit RBMs and NADE are taken from [11]. $\pm$ indicates a confidence interval of three standard deviations.

partition to monitor model performance on a held-out validation set. When the onset of over-fitting is detected, we store the model parameters and report the associated test-set likelihood, as estimated by both AIS and our tracking algorithm. The advantages of such an early-stopping procedure is shown in Figure 4(b), where training log-likelihood increases throughout training while validation performance starts to decrease around 250 epochs. Detecting over-fitting without tracking the log partition would require a dense grid of AIS runs which would prove computationally prohibitive.

We tested parameters in the following range: number of hidden units in $\{100, 200, 500, 1000\}$ (depending on dataset size), learning rates in $\{10^{-2}, 10^{-3}, 10^{-4}\}$ either held constant during training or annealed with constants $\alpha \in \{10^3, 10^4, 10^5\}$. For tempering, we used 10 fixed temperatures, spaced linearly between $\beta = [0, 1]$. SGD was performed using mini-batches of size $\{10, 100\}$ when estimating the gradient, and mini-batches of size $\{10, 20\}$ for our set of tempered-chains (we thus simulate $10 \times \{10, 20\}$ tempered chains in total). As can be seen in Table 4, our tracker performs very well compared to the AIS estimates and across all datasets. Efforts to lower the variance of the AIS estimate proved unsuccessful, even going as far as $10^5$ intermediate distributions.

## 5 Discussion

In this paper, we have shown that while exact calculation of the partition function of RBMs may be intractable, one can exploit the smoothness of gradient descent learning in order to approximately track the evolution of the log partition function during learning. Treating the $\zeta_{i,t}$'s as latent variables, the graphical model of Figure 1 allowed us to combine multiple sources of information to achieve good tracking of the log partition function throughout training, on a variety of datasets. We note however that good tracking performance is contingent on the ergodicity of the negative phase sampler. Unsurprisingly, this is the same condition required by SML for accurate estimation of the negative phase gradient.

The method presented in the paper is also computationally attractive, with only a small computaiton overhead relative to SML-PT training. The added computational cost lies in the computation of the importance weights for importance sampling and bridge sampling. However, this boils down to computing free-energies which are mostly pre-computed in the course of gradient updates with the sole exception being the computation of $\tilde{q}_{i,t}(x_{i,t-1})$ in the importance sampling step. In comparison to AIS, our method allows us to fairly accurately track the log partition function, and at a per-point estimate cost well below that of AIS. Having a reliable and accurate online estimate of the log partition function opens the door to a wide range of new research directions.

**Acknowledgments**

The authors acknowledge the financial support of NSERC and CIFAR; and Calcul Québec for computational resources. We also thank Hugo Larochelle for access to the datasets of Sec. 4; Hannes Schulz, Andreas Mueller, Olivier Delalleau and David Warde-Farley for feedback on the paper and algorithm; along with the developers of Theano [3].

## Footnotes

[1]Stochastic gradient descent is one of the most popular methods for training MRFs precisely because second order optimization methods typically require a deterministic gradient, whereas sampling-based estimators are the only practical option for models with an intractable partition function.

[2] This same technique was recently used in [5], in the context of learning rate adaptation.

[3] The visible units of an RBM with zero weights are marginally independent. Its log partition function is thus given as $\sum_i \log(1 + \exp(b_i)) + n_h \cdot \log(2)$.

[4]Each $b_k$ is initialized to $\log \frac{\bar{x}_k}{1-\bar{x}_k}$, where $\bar{x}_k$ is the mean of the $k$-th dimension on the training set.

[5]Our base AIS config. was $10^3$ intermediate distributions spaced linearly between $\beta = [0, 0.5]$, $10^4$ distributions for the interval $[0.5, 0.9]$ and $10^4$ for $[0.9, 1.0]$. Estimates of $\log Z$ are averaged over 100 annealed importance weights.

# References

[1] Bengio, Y. (2009). Learning deep architectures for AI. *Foundations and Trends in Machine Learning*, **2**(1), 1–127. Also published as a book. Now Publishers, 2009.

[2] Bennett, C. (1976). Efficient estimation of free energy differences from Monte Carlo data. *Journal of Computational Physics*, **22**(2), 245–268.

[3] Bergstra, J., Breuleux, O., Bastien, F., Lamblin, P., Pascanu, R., Desjardins, G., Turian, J., Warde-Farley, D., and Bengio, Y. (2010). Theano: a CPU and GPU math expression compiler. In *Proceedings of the Python for Scientific Computing Conference (SciPy)*. Oral.

[4] Bishop, C. M. (2006). *Pattern Recognition and Machine Learning*. Springer.

[5] Cho, K., Raiko, T., and Ilin, A. (2011). Enhanced gradient and adaptive learning rate for training restricted boltzmann machines. In L. Getoor and T. Scheffer, editors, *Proceedings of the 28th International Conference on Machine Learning (ICML-11)*, ICML '11, pages 105–112, New York, NY, USA. ACM.

[6] Desjardins, G., Courville, A., and Bengio, Y. (2010a). Adaptive parallel tempering for stochastic maximum likelihood learning of rbms. *NIPS*2010 Deep Learning and Unsupervised Feature Learning Workshop*.

[7] Desjardins, G., Courville, A., Bengio, Y., Vincent, P., and Delalleau, O. (2010b). Tempered Markov chain monte carlo for training of restricted Boltzmann machine. In *JMLR W&CP: Proceedings of the Thirteenth International Conference on Artificial Intelligence and Statistics (AISTATS 2010)*, volume 9, pages 145–152.

[8] Hinton, G. (2010). A practical guide to training restricted boltzmann machines. Technical Report 2010003, University of Toronto. version 1.

[9] Hinton, G. E., Osindero, S., and Teh, Y. (2006). A fast learning algorithm for deep belief nets. *Neural Computation*, **18**, 1527–1554.

[10] Iba, Y. (2001). Extended ensemble monte carlo. *International Journal of Modern Physics*, **C12**, 623–656.

[11] Larochelle, H. and Murray, I. (2011). The Neural Autoregressive Distribution Estimator. In *Proceedings of the Fourteenth International Conference on Artificial Intelligence and Statistics (AISTATS'2011)*, volume 15 of JMLR: W&CP.

[12] Larochelle, H., Bengio, Y., and Turian, J. (2010). Tractable multivariate binary density estimation and the restricted Boltzmann forest. *Neural Computation*, **22**(9), 2285–2307.

[13] LeCun, Y., Bottou, L., Bengio, Y., and Haffner, P. (1998). Gradient based learning applied to document recognition. *IEEE*, **86**(11), 2278–2324.

[14] Lingenheil, M., Denschlag, R., Mathias, G., and Tavan, P. (2009). Efficiency of exchange schemes in replica exchange. *Chemical Physics Letters*, **478**(1-3), 80 – 84.

[15] Marinari, E. and Parisi, G. (1992). Simulated tempering: A new monte carlo scheme. *EPL (Europhysics Letters)*, **19**(6), 451.

[16] Marlin, B., Swersky, K., Chen, B., and de Freitas, N. (2009). Inductive principles for restricted boltzmann machine learning. In *Proceedings of The Thirteenth International Conference on Artificial Intelligence and Statistics (AISTATS'10)*, volume 9, pages 509–516.

[17] Murray, I. and Ghahramani, Z. (2004). Bayesian learning in undirected graphical models: Approximate mcmc algorithms.

[18] Neal, R. M. (2001). Annealed importance sampling. *Statistics and Computing*, **11**(2), 125–139.

[19] Neal, R. M. (2005). Estimating ratios of normalizing constants using linked importance sampling.

[20] Salakhutdinov, R. (2010a). Learning deep boltzmann machines using adaptive mcmc. In L. Bottou and M. Littman, editors, *Proceedings of the Twenty-seventh International Conference on Machine Learning (ICML-10)*, volume 1, pages 943–950. ACM.

[21] Salakhutdinov, R. (2010b). Learning in Markov random fields using tempered transitions. In *NIPS'09*.

[22] Salakhutdinov, R. and Hinton, G. E. (2009). Deep Boltzmann machines. In *AISTATS'2009*, volume 5, pages 448–455.

[23] Salakhutdinov, R. and Murray, I. (2008). On the quantitative analysis of deep belief networks. In W. W. Cohen, A. McCallum, and S. T. Roweis, editors, *ICML 2008*, volume 25, pages 872–879. ACM.

[24] Taylor, G. and Hinton, G. (2009). Factored conditional restricted Boltzmann machines for modeling motion style. In L. Bottou and M. Littman, editors, *ICML 2009*, pages 1025–1032. ACM.

[25] Tieleman, T. (2008). Training restricted Boltzmann machines using approximations to the likelihood gradient. In W. W. Cohen, A. McCallum, and S. T. Roweis, editors, *ICML 2008*, pages 1064–1071. ACM.

[26] Tieleman, T. and Hinton, G. (2009). Using fast weights to improve persistent contrastive divergence. In L. Bottou and M. Littman, editors, *ICML 2009*, pages 1033–1040. ACM.

[27] Welling, M., Rosen-Zvi, M., and Hinton, G. E. (2005). Exponential family harmoniums with an application to information retrieval. In *NIPS'04*, volume 17, Cambridge, MA. MIT Press.

